# The Value of Labeled and Unlabeled Examples when the Model is Imperfect

**Kaushik Sinha**
Dept. of Computer Science and Engineering
Ohio State University
Columbus, OH 43210
sinhak@cse.ohio-state.edu

**Mikahil Belkin**
Dept. of Computer Science and Engineering
Ohio State University
Columbus, OH 43210
mbelkin@cse.ohio-state.edu

## Abstract

Semi-supervised learning, i.e. learning from both labeled and unlabeled data has received significant attention in the machine learning literature in recent years. Still our understanding of the theoretical foundations of the usefulness of unlabeled data remains somewhat limited. The simplest and the best understood situation is when the data is described by an identifiable mixture model, and where each class comes from a pure component. This natural setup and its implications ware analyzed in [11, 5]. One important result was that in certain regimes, labeled data becomes exponentially more valuable than unlabeled data.

However, in most realistic situations, one would not expect that the data comes from a parametric mixture distribution with identifiable components. There have been recent efforts to analyze the non-parametric situation, for example, "cluster" and "manifold" assumptions have been suggested as a basis for analysis. Still, a satisfactory and fairly complete theoretical understanding of the nonparametric problem, similar to that in [11, 5] has not yet been developed.

In this paper we investigate an intermediate situation, when the data comes from a probability distribution, which can be modeled, but not perfectly, by an identifiable mixture distribution. This seems applicable to many situation, when, for example, a mixture of Gaussians is used to model the data. the contribution of this paper is an analysis of the role of labeled and unlabeled data depending on the amount of imperfection in the model.

## 1 Introduction

In recent years semi-supervised learning, i.e. learning from labeled and unlabeled data, has drawn significant attention. The ubiquity and easy availability of unlabeled data together with the increased computational power of modern computers, make the paradigm attractive in various applications, while connections to natural learning make it also conceptually intriguing. See [15] for a survey on semi-supervised learning.

From the theoretical point of view, semi-supervised learning is simple to describe. Suppose the data is sampled from the joint distribution $p(x, y)$, where $x$ is a feature and $y$ is the label. The unlabeled data comes from the marginal distribution $p(x)$. Thus the the usefulness of unlabeled data is tied to how much information about joint distribution can be extracted from the marginal distribution. Therefore, in order to make unlabeled data useful, an assumption on the connection between these distributions needs to be made.

In the non-parametric setting several such assumptions have been recently proposed, including the the cluster assumption and its refinement, the low-density separation assumption [7, 6], and the manifold assumption [3]. These assumptions relate the shape of the marginal probability distribution to class labels. The low-density separation assumption states that the class boundary passes through the low density regions, while the manifold assumption proposes that the proximity of the points should be measured along the data manifold. However, while these assumptions has motivated several algorithms and have been shown to hold empirically, few theoretical results on the value of unlabeled data in the non-parametric setting are available so far. We note the work of Balcan and Blum ([2]), which attempts to unify several frameworks by introducing a notion of compatibility between labeled and unlabeled data. In a slightly different setting some theoretical results are also available for co-training ([4, 8]).

Far more complete results are available in the parametric setting. There one assumes that the distribution $p(x, y)$ is a mixture of two parametric distribution $p_1$ and $p_2$, each corresponding to a different class. Such mixture is called *identifiable*, if parameters of each component can be uniquely determined from the marginal distribution $p(x)$. The study of usefulness of unlabeled data under this assumption was undertaken by Castelli and Cover ([5]) and Ratsaby and Venkatesh ([11]). Among several important conclusions from their study was the fact under a certain range of conditions, labeled data is exponentially more important for approximating the Bayes optimal classifier than unlabeled data. Roughly speaking, unlabeled data may be used to identify the parameters of each mixture component, after which the class attribution can be established exponentially fast using only few labeled examples.

While explicit mixture modeling is of great theoretical and practical importance, in many applications there is no reason to believe that the model provides a precise description of the phenomenon. Often it is more reasonable to think that our models provide a rough approximation to the underlying probability distribution, but do not necessarily represent it exactly. In this paper we investigate the limits of usefulness of unlabeled data as a function of how far the best fitting model strays from the underlying probability distribution.

The rest of the paper is structured as follows: we start with an overview of the results available for identifiable mixture models together with some extensions of these results. We then describe how the relative value of labeled and unlabeled data changes when the true distribution is a perturbation of a parametric model. Finally we discuss various regimes of usability for labeled and unlabeled data and represent our findings in Fig 1.

## 2 Relative Value of Labeled and Unlabeled Examples

Our analysis is conducted in the standard classification framework and studies the behavior $P_{error} - P_{Bayes}$, where $P_{error}$ is probability of misclassification for a given classifier and $P_{Bayes}$ is the classification error of the optimal classifier. The quantity $P_{error} - P_{Bayes}$ is often referred to as *the excess probability of error* and expresses how far our classifier is from the best possible.

In what follows, we review some theoretical results that describe behavior of the excess error probability as a function of the number of labeled and unlabeled examples. We will denote number of labeled examples by $l$ and the number of unlabeled examples by $u$. We omit certain minor technical details to simplify the exposition. The classifier, for which $P_{error}$ is computed is based on the underlying model.

**Theorem 2.1. (Ratsaby and Venkatesh [11])** *In a two class identifiable mixture model, let the equiprobable class densities $p_1(x), p_2(x)$ be $d$-dimensional Gaussians with unit covariance matrices. Then for sufficiently small $\epsilon > 0$ and arbitrary $\delta > 0$, given $l = \mathcal{O}\left(\log \delta^{-1}\right)$ labeled and $u = \mathcal{O}\left(\frac{d^2}{\epsilon^3 \delta}(d \log \epsilon^{-1} + \log \delta^{-1})\right)$ unlabeled examples respectively, with confidence at least $1 - \delta$, probability of error $P_{error} \leq P_{Bayes}(1 + c\epsilon)$ for some positive constant $c$.*

Since the mixture is identifiable, parameters can be estimated from unlabeled examples alone. Labeled examples are not required for this purpose. Therefore, unlabeled examples are used to estimate the mixture and hence the two decision regions. Once the decision regions are established, labeled examples are used to label them. An equivalent form of the above result in terms of labeled and unlabeled examples is $P_{error} - P_{Bayes} = \mathcal{O}\left(\frac{d}{u^{1/3}}\right) + \mathcal{O}\left(\exp(-l)\right)$. For a fixed dimension $d$, this

indicates that labeled examples are exponentially more valuable than the unlabeled examples in reducing the excess probability of error, however, when $d$ is not fixed, higher dimensions slower these rates.

Independently, Cover and Castelli provided similar results in a different setting under Bayesian framework.

**Theorem 2.2. (Cover and Castelli [5])** *In a two class mixture model, let $p_1(x), p_2(x)$ be the parametric class densities and let $h(\eta)$ be the prior over the unknown mixing parameter $\eta$. Then*

$$P_{error} - P_{Bayes} = \mathcal{O}\left(\frac{1}{u}\right) + \exp\{-Dl + o(l)\}$$

*where $D = -\log\{2\sqrt{\eta(1-\eta)} \int \sqrt{p_1(x)p_2(x)}dx\}$*

In their framework, Cover and Castelli [5] assumed that parameters of individual class densities are known, however the associated class labels and mixing parameter are unknown. Under such assumption their result shows that the above rate is obtained when $l^{3+\epsilon}u^{-1} \to 0$ as $l + u \to \infty$. In particular this implies that, if $ue^{-Dl} \to 0$ and $l = o(u)$ the excess error is essentially determined by the number of unlabeled examples. On the other hand if $u$ grows faster than $e^{Dl}$, then excess error is determined by the number of labeled examples. For detailed explanation of the above statements see pp-2103 [5]. The effect of dimensionality is not captured in their result.

Both results indicate that if the parametric model assumptions are satisfied, labeled examples are exponentially more valuable than unlabeled examples in reducing the excess probability of error.

In this paper we investigate the situation when the parametric model assumptions are only satisfied to a certain degree of precision, which seems to be a natural premise in a variety of practical settings.

It is interesting to note that uncertainty can appear for different reasons. One source of uncertainty is a lack of examples, which we call Type-A. Imperfection of the model is another source of uncertainty, which we will refer to as Type-B.

- **Type-A uncertainty for perfect model with imperfect information:** Individual class densities follow the assumed parametric model. Uncertainty results from finiteness of examples. *Perturbation size* specifies how well parameters of the individual class densities can be estimated from finite data.

- **Type-B uncertainty for imperfect model:** Individual class densities does not follow the assumed parametric model. *Perturbation size* specifies how well the best fitting model can approximate the underlying density.

Before proceeding further, we describe our model and notations. We take the instance space $X \subset \mathbb{R}^d$ with labels $\{-1, 1\}$. True class densities are always represented by $p_1(x)$ and $p_2(x)$ respectively. In case of Type-A uncertainty they are simply $p_1(x|\theta_1)$ and $p_2(x|\theta_2)$. In case of Type-B uncertainty $p_1(x), p_2(x)$ are perturbations of two $d$-dimensional densities from a parametric family $\mathcal{F}$. We will denote the mixing parameter by $t$ and the individual parametric class densities by $f_1(x|\theta_1), f_2(x|\theta_2)$ respectively and the resulting mixture density as $tf_1(x|\theta_1) + (1-t)f_2(x|\theta_2)$. We will show some specific results when $\mathcal{F}$ consists of spherical Gaussian distributions with unit covariance matrix and $t = \frac{1}{2}$. In such a case $\theta_1, \theta_2 \in \mathbb{R}^d$ represent the means of the corresponding densities and the mixture density is indexed by a $2d$ dimensional vector $\theta = [\theta_1, \theta_2]$. The class of such mixtures is identifiable and hence using unlabeled examples alone, $\theta$ can be estimated by $\hat{\theta} \in \mathbb{R}^{2d}$. By $|| \cdot ||$ we represent the standard Euclidean norm in $\mathbb{R}^d$ and by $|| \cdot ||_{\frac{d}{2},2}$ the Sobolev norm. Note that for some $\epsilon > 0$, $|| \cdot ||_{\frac{d}{2},2} < \epsilon$ implies $|| \cdot ||_1 < \epsilon$ and $|| \cdot ||_\infty < \epsilon$. We will frequently use the following term $L(a,t,e) = \frac{\log(\frac{a}{\delta})}{(t-Ae)(1-2\sqrt{(P_{Bayes}+Be)(1-P_{Bayes}-Be)})}$ to represent the optimal number of labeled examples for correctly classifying estimated decision regions with high probability (as will be clear in the next section) where, $t$ represents mixing parameter, $e$ represents perturbation size and $a$ is an integer variable and $A, B$ are constants.

## 2.1 Type-A Uncertainty : Perfect Model Imperfect Information

Due to finiteness of unlabeled examples, density parameters can not be estimated arbitrarily close to the true parameters in terms of Euclidean norm. Clearly, how close they can be estimated depends on the number of unlabeled examples used $u$, dimension $d$ and confidence probability $\delta$. Thus, Type-I uncertainty inherently gives rise to a perturbation size defined by $\epsilon_1(u, d, \delta)$ such that, a fixed $u$ defines a perturbation size $\epsilon_1(d, \delta)$. Because of this perturbation, estimated decision regions differ from the true decision regions. From [11] it is clear that only very few labeled examples are good enough to label these two estimated decision regions reasonably well with high probability. Let such a number of labeled examples be $l^*$. But what happens if the number of labeled examples available is greater than $l^*$? Since the individual densities follow the parametric model exactly, these extra labeled examples can be used to estimate the density parameters and hence the decision regions. However, using a simple union bound it can be shown ([10]) that the asymptotic rate for convergence of such estimation procedure is $\mathcal{O}\left(\sqrt{\frac{d}{l}\log(\frac{d}{\delta})}\right)$. Thus, provided we have $u$ unlabeled examples if we want to represent the rate at which excess probability of error reduces as a function of the number of labeled examples, it is clear that initially the error reduces exponentially fast in number of labeled examples (following [11]) but then it reduces only at a rate $\mathcal{O}\left(\sqrt{\frac{d}{l}\log(\frac{d}{\delta})}\right)$. Provided we use the following strategy, this extends the result of [11] as given in the Theorem below.

We adopt the following strategy to utilize labeled and unlabeled examples in order to learn a classification rule.
**Strategy 1:**

1. Given $u$ unlabeled examples, and confidence probability $\delta > 0$ use maximum likelihood estimation method to learn the parameters of the mixture model such that the estimates $\hat{\theta}_1, \hat{\theta}_2$ are only $\epsilon_1(u, d, \delta) = \mathcal{O}^*\left(\frac{d}{u^{1/3}}\right)$ close to the actual parameters with probability at least $1 - \frac{\delta}{4}$.

2. Use $l^*$ labeled examples to label the estimated decision regions with probability of incorrect labeling no greater than $\frac{\delta}{4}$.

3. If $l > l^*$ examples are available use them to estimate the individual density parameters with probability at least $1 - \frac{\delta}{2}$.

**Theorem 2.3.** *Let the model be a mixture of two equiprobable $d$ dimensional spherical Gaussians $p_1(x|\theta_1), p_2(x|\theta_2)$ having unit covariance matrices and means $\theta_1, \theta_2 \in \mathbb{R}^d$. For any arbitrary $1 > \delta > 0$, if strategy 1 is used with $u$ unlabeled examples then there exists a perturbation size $\epsilon_1(u, d, \delta) > 0$ and positive constants $A, B$ such that using $l \leq l^* = L(24, 0.5, \epsilon_1)$ labeled examples, $P_{error} - P_{Bayes}$ reduces exponentially fast in the number of labeled examples with probability at least $(1 - \frac{\delta}{2})$. If more labeled examples $l > l^*$ are provided then with probability at least $(1 - \frac{\delta}{2})$, $P_{error} - P_{Bayes}$ asymptotically converges to zero at a rate $\mathcal{O}\left(\sqrt{\frac{d}{l}\log(\frac{d}{\delta})}\right)$ as $l \to \infty$. If we represent the reduction rate of this excess error$(P_{error} - P_{Bayes})$ as a function of labeled examples $R_{ee}(l)$, then this can be compactly represented as,*

$$R_{ee}(l) = \begin{cases} \mathcal{O}\left(\exp(-l)\right) & \text{if } l \leq l^* \\ \mathcal{O}\left(\sqrt{\frac{d}{l}\log(\frac{d}{\delta})}\right) & \text{if } l > l^* \end{cases}$$

After using $l^*$ labeled examples $P_{error} = P_{Bayes} + \mathcal{O}(\epsilon_1)$.

## 2.2 Type-B Uncertainty: Imperfect Model

In this section we address the main question raised in this paper. Here the individual class densities do not follow the assumed parametric model exactly but are a perturbed version of the assumed model. The uncertainty in this case is specified by the perturbation size $\epsilon_2$ which roughly indicates by what extent the true class densities differ form that of the best fitting parametric model densities.

For any mixing parameter $t \in (0, 1)$ let us consider a two class mixture model with individual class densities $p_1(x), p_2(x)$ respectively. Suppose the best knowledge available about this mixture model is that individual class densities approximately follow some parametric form from a class $\mathcal{F}$. We assume that best approximations of $p_1, p_2$ within $\mathcal{F}$ are $f_1(x|\theta_1), f_2(x|\theta_2)$ respectively, such that for $i \in \{1, 2\}$, $(f_i - p_i)$ are in Sobolev class $H^{\frac{d}{2}}$ and there exists a perturbation size $\epsilon_2 > 0$ such that $||p_1 - f_1||_{\frac{d}{2}, 2} \leq \epsilon_2$ and $||p_2 - f_2||_{\frac{d}{2}, 2} \leq \epsilon_2$. Here, the Sobolev norm is used as a smoothness condition and implies that true densities are smooth and not "too different" from the best fitting parametric model densities and in particular, if $||f_i - p_i||_{\frac{d}{2}, 2} \leq \epsilon_2$ then $||f_i - p_i||_1 \leq \epsilon_2$ and $||f_i - p_i||_\infty \leq \epsilon_2$.

We first show that due to the presence of this perturbation size, even complete knowledge of the best fitting model parameters does not help in learning optimal classification rule in the following sense. In the absence of any perturbation, complete knowledge of model parameters implies that the decision boundary and hence two decision regions are explicitly known but not their labels. Thus, using only a very small number of labeled examples $P_{error}$ reduces exponentially fast in the number of labeled examples to $P_{Bayes}$ as number of labeled examples increases. However, due to the presence of perturbation size, $P_{error}$ reduces exponentially fast in number of labeled examples only up to $P_{Bayes} + \mathcal{O}(\epsilon_2)$. Since beyond this, parametric model assumptions do not hold due to the presence of perturbation size, some non parametric technique must be used to estimate the actual decision boundary. For any such nonparametric technique $P_{error}$ now reduces at a much slower rate. This trend is roughly what the following theorem says. Here $f_1, f_2$ are general parametric densities not necessarily Gaussians. In what follows we assume that $p_1, p_2 \in \mathcal{C}^\infty$ and hence convergence rate for non parametric classification (see [14]) is $\mathcal{O}\left(\frac{1}{\sqrt{l}}\right)$. Slower rate results if infinite differentiability condition is not satisfied.

**Theorem 2.4.** *In a two class mixture model with individual class densities $p_1(x), p_2(x)$ and mixing parameter $t \in (0, 1)$, let the mixture density of best fitting parametric model be $t f_1(x|\theta_1) + (1 - t) f_2(x|\theta_2)$ where $f_1, f_2$ belongs to some parametric class $\mathcal{F}$ and true densities $p_1, p_2$ are perturbed version of $f_1, f_2$. For a perturbation size $\epsilon_2 > 0$, if $||f_1 - p_1||_{\frac{d}{2}, 2} \leq \epsilon_2, ||f_2 - p_2||_{\frac{d}{2}, 2} \leq \epsilon_2$ and $\theta_1, \theta_2$ are known then for any $0 < \delta < 1$, there exists positive constants $A, B$ such that for $l \leq l^* = L(6, t, \epsilon_2)$ labeled example, $P_{error} - P_{Bayes}$ reduces exponentially fast in the number of labeled examples with probability at least $(1 - \delta)$. If more labeled examples $l > l^*$ are provided $P_{error} - P_{Bayes}$ asymptotically converges to zero at a rate $\mathcal{O}\left(\frac{1}{\sqrt{l}}\right)$ as $l \to \infty$.*

After using $l^*$ labeled examples $P_{error} = P_{Bayes} + \mathcal{O}(\epsilon_2)$. Thus, from the above theorem it can be thought that as labeled examples are added, initially the excess error reduces at a very fast rate (exponentially in the number of labeled examples) until $P_{error} - P_{Bayes} = \mathcal{O}(\epsilon_2)$. After that the excess error reduces only polynomially fast in the number of labeled examples. In proving of the above theorem we used first order Taylor series approximation to get an crude upper bound for decision boundary movement. However, in case of a specific class of parametric densities such a crude approximation may not be necessary. In particular, as we show next, if the best fitting model is a mixture of spherical Gaussians where the boundary is linear hyperplane, explicit upper bound of boundary movement can be found. In the following, we assume the class $\mathcal{F}$ to be a class of $d$ dimensional spherical Gaussians with identity covariance matrix. However, the true model is an equiprobable mixture of perturbed versions of these individual class densities. As before, given $u$ unlabeled examples and $l$ labeled examples we want a strategy to learn a classification rule and analyze the effect of these examples and also of perturbation size $\epsilon_2$ in reducing excess probability of error.

One option to achieve this task is to use the unlabeled examples to estimate the true mixture density $\frac{1}{2}p_1 + \frac{1}{2}p_2$, however number of unlabeled examples required to estimate mixture density using non parametric kernel density estimation is exponential to the number of dimensions [10]. Thus, for high dimensional data this is not an attractive option and also such an estimate does not provide any clue as to where the decision boundary is. A better option will be to use the unlabeled examples to estimate the best fitting Gaussians within $\mathcal{F}$. Number of unlabeled examples needed to estimate such a mixture of Gaussians is only polynomial in the number of dimensions [10] and it is easy to show that the distance between the Bayesian decision function and the decision function due to Gaussian approximation is at most $\epsilon_2$ away in $||.||_{\frac{d}{2}, 2}$ norm sense.

Now suppose we use the following strategy to use labeled and unlabeled examples.
**Strategy 2:**

1. Assume the examples are distributed according to a mixture of equiprobable Gaussians with unit covariance matrices and apply maximum likelihood estimation method to find the best Gaussian approximation of the densities.

2. Use small number of labeled examples $l^*$ to label the two approximate decision regions correctly with high probability.

3. If more $(l > l^*)$ labeled examples are available, use them to learn a better decision function using some nonparametric technique.

**Theorem 2.5.** *In a two class mixture model with equiprobable class densities $p_1(x), p_2(x)$, let the mixture density of the best fitting parametric model be $\frac{1}{2}f_1(x|\theta_1) + \frac{1}{2}f_2(x|\theta_2)$ where $f_1, f_2$ are $d$ dimensional spherical Gaussians with means $\theta_1, \theta_2 \in \mathbb{R}^d$ and $p_1, p_2$ are perturbed version of $f_1, f_2$, such that for a perturbation size $\epsilon_2 > 0$, $\|f_1 - p_1\|_{\frac{d}{2},2} \leq \epsilon_2, \|f_2 - p_2\|_{\frac{d}{2},2} \leq \epsilon_2$. For any $\epsilon > 0$ and $0 < \delta < 1$, there exists positive constants $A, B$ such that if strategy 2 is used with $u = \mathcal{O}\left(\frac{d^2}{\epsilon^3 \delta}(d \log \frac{1}{\epsilon} + \log \frac{1}{\delta})\right)$ unlabeled and $l^* = L(0.5, 12, (\epsilon + \epsilon_2))$ labeled examples then for $l \leq l^*$, $P_{error} - P_{Bayes}$ reduces exponentially fast in the number of labeled examples with probability at least $(1 - \delta)$. If more labeled examples $l > l^*$ are provided, $P_{error} - P_{Bayes}$ asymptotically converges to zero at most at a rate $\mathcal{O}\left(\frac{1}{\sqrt{l}}\right)$ as $l \to \infty$. If we represent the reduction rate of this excess error $(P_{error} - P_{Bayes})$ as a function of labeled examples as $R_{ee}(l)$, then this can compactly represented as,*

$$R_{ee}(l) = \begin{cases} \mathcal{O}\left(\exp(-l)\right) & \text{if } l \leq l^* \\ \mathcal{O}\left(\frac{1}{\sqrt{l}}\right) & \text{if } l > l^* \end{cases}$$

After using $l^*$ labeled examples, $P_{error} = P_{Bayes} + \mathcal{O}(\epsilon + \epsilon_2)$. Note that when number of unlabeled examples is infinite, parameters of the best fitting model can be estimated arbitrarily well, i.e., $\epsilon \to 0$ and hence $P_{error} - P_{Bayes}$ reduces exponentially fast in the number of labeled examples until $P_{error} - P_{Bayes} = \mathcal{O}(\epsilon_2)$. On the other hand if $\epsilon = \mathcal{O}(\epsilon_2)$, $P_{error} - P_{Bayes}$ still reduces exponentially fast in the number of labeled examples until $P_{error} - P_{Bayes} = \mathcal{O}(\epsilon_2)$. This implies that $\mathcal{O}(\epsilon_2)$ close estimate of parameters of the best fitting model is "good" enough. A more precise estimate of parameters of the best fitting model using more unlabeled examples does not help reducing $P_{error} - P_{Bayes}$ at the same exponential rate beyond $P_{error} - P_{Bayes} = \mathcal{O}(\epsilon_2)$. The following Corollary states this important fact.

**Corollary 2.6.** *For a perturbation size $\epsilon_2 > 0$, let the best fitting model for a mixture of equiprobable densities be a mixture of equiprobable $d$ dimensional spherical Gaussians with unit covariance matrices. If using $u^* = \mathcal{O}\left(\frac{d^2}{\epsilon_2^3 \delta}(d \log \frac{1}{\epsilon_2} + \log \frac{1}{\delta})\right)$ unlabeled examples parameters of the best fitting model can be estimated $\mathcal{O}(\epsilon_2)$ close in Euclidean norm sense, then any additional unlabeled examples $u > u^*$ does not help in reducing the excess error.*

## 3 Discussion on different rates of convergence

In this section we discuss the effect of perturbation size $\epsilon_2$ on the behavior of $P_{error} - P_{Bayes}$ and its effect on controlling the value of labeled and unlabeled examples. Different combinations of number of labeled and unlabeled examples give rise to four different regions where $P_{error} - P_{Bayes}$ behaves differently as shown in Figure 1 where the $x$ axis corresponds to the number of unlabeled examples and the $y$ axis corresponds to the number of labeled examples.

Let $u^*$ be the number of unlabeled examples required to estimate the parameters of the best fitting model $\mathcal{O}(\epsilon_2)$ close in Euclidean norm sense. Using $\mathcal{O}^*$ notation to hide the log factors, according to Theorem 2.5, $u^* = \mathcal{O}^*\left(\frac{d^3}{\epsilon_2^3}\right)$. When $u > u^*$, unlabeled examples have no role to play in reducing $P_{error} - P_{Bayes}$ as shown in region II and part of III in Figure 1. For $u \leq u^*$, unlabeled examples becomes useful only in region I and IV. When $u^*$ unlabeled examples are available to estimate the parameters of the best fitting model $\mathcal{O}(\epsilon_2)$ close, let the number of labeled examples required to

label the estimated decision regions so that $P_{error} - P_{Bayes} = \mathcal{O}(\epsilon_2)$ be $l^*$. The figure is just for graphical representation of different regions where $P_{error} - P_{Bayes}$ reduces at different rates.

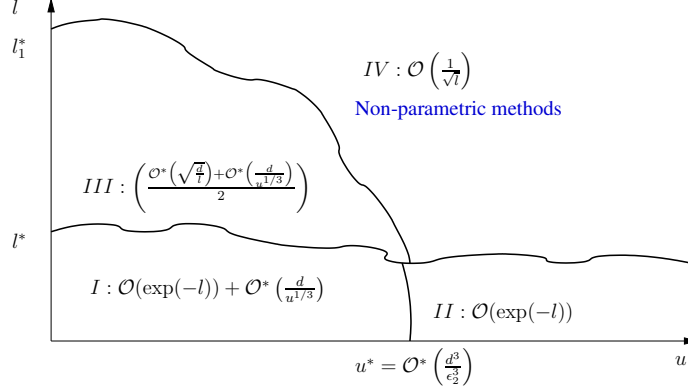

Figure 1: **The Big Picture.** Behavior of $P_{error} - P_{Bayes}$ for different labeled and unlabeled examples

### 3.1 Behavior of $P_{error} - P_{Bayes}$ in Region-I

In this region $u \leq u^*$ unlabeled examples estimate the decision regions and $l_u^*$ labeled examples, which depends on $u$, are required to correctly label these estimated regions. $P_{error} - P_{Bayes}$ reduces at a rate $\mathcal{O}\left(\exp(-l)\right) + \mathcal{O}\left(\frac{d}{u^{\frac{1}{3}}}\right)$ for $u < u^*$ and $l < l_u^*$. This rate can be interpreted as the rate at which unlabeled examples estimate the parameters of the best fitting model and rate at which labeled examples correctly label these estimated decision regions. However, for small $u$ estimation of the decision regions will be bad and and corresponding $l_u^* > l^*$. Instead of using these large number labeled examples to label poorly estimated decision regions, they can instead be used to estimate the parameters of the best fitting model and as will be seen next, this is precisely what happens in region III. Thus in region I, $l$ is restricted to $l < l^*$ and $P_{error} - P_{Bayes}$ reduces at a rate $\exp\left(-\mathcal{O}(l)\right) + \mathcal{O}\left(\frac{d}{u^{\frac{1}{3}}}\right)$.

### 3.2 Behavior of $P_{error} - P_{Bayes}$ in Region-II

In this section $l \leq l^*$ and $u > u^*$. As shown in Corollary 2.6, using $u^*$ unlabeled examples parameters of the best fitting model can be estimated $\mathcal{O}(\epsilon_2)$ close in Euclidean norm sense and more precise estimate of the best fitting model parameters using more unlabeled examples $u > u^*$ does not help reducing $P_{error} - P_{Bayes}$. Thus, unlabeled examples have no role to play in this region and for small number of labeled examples $l \leq l^*$, $P_{error} - P_{Bayes}$ reduces at a rate $\mathcal{O}\left(\exp(-l)\right)$.

### 3.3 Behavior of $P_{error} - P_{Bayes}$ in Region-III

In this region $u \leq u^*$ and hence model parameters have not been estimated $\mathcal{O}(\epsilon_2)$ close to the parameters of the best fitting model. Thus, in some sense model assumptions are still valid and there is a scope for better estimation of the parameters. Number of labeled examples available in this region is greater than what is required for mere labeling the estimated decision regions using $u$ unlabeled examples and hence these excess labeled examples can be used to estimate the model parameters. Note that once the parameters have been estimated $\mathcal{O}(\epsilon_2)$ close to the parameters of the best fitting model using labeled examples, parametric model assumptions are no longer valid. If $l_1^*$ is the number of such labeled examples, then in this region $l^* < l \leq l_1^*$. Also note that depending on number of unlabeled examples $u \leq u^*$, $l^*$, and $l_1^*$ are not fixed numbers but will depend on $u$. In presence of labeled examples alone, using Theorem 2.3, $P_{error} - P_{Bayes}$ reduces at a rate $\mathcal{O}^*\left(\sqrt{\frac{d}{l}}\right)$. Since parameters are being estimated both using labeled and unlabeled examples, the

effective rate at which $P_{error} - P_{Bayes}$ reduces at this region can be thought of as the mean of the two.

### 3.4    Behavior of $P_{error} - P_{Bayes}$ in Region-IV

In this region when $u > u^*, l > l^*$ and when $u \leq u^*, l > l_1^*$. In either case, since the parameters of the best fitting model have been estimated $\mathcal{O}(\epsilon_2)$ close to the parameters of the best fitting model, parametric model assumptions are also no longer valid and excess labeled examples must be used in nonparametric way. For nonparametric classification technique either one of the two basic families of classifiers, plug-in classifiers or empirical risk minimization (ERM) classifiers can be used [13, 9]. A nice discussion on the rate and fast rate of convergence of both these types of classifiers can be found in [1, 12]. The general convergence rate i.e. the rate at which expected value of $(P_{error} - P_{Bayes})$ reduces is of the order $\mathcal{O}(l^{-\beta})$ as $l \to \infty$ where $\beta > 0$ is some exponent and is typically $\beta \leq 0.5$. Also it was shown in [14] that under general conditions this bound can not be improved in a minimax sense. In particular it was shown that if the true densities belong to $\mathcal{C}^\infty$ class then this rate is $\mathcal{O}(\frac{1}{\sqrt{l}})$. However, if infinite differentiability condition is not satisfied then this rate is much slower.

**Acknowledgements** This work was supported by NSF Grant No 0643916.

## References

[1] J. Y. Audibert and A. Tsybakov. Fast convergence rate for plug-in estimators under margin conditions. In *Unpublished manuscript*, 2005.

[2] M-F. Balcan and A. Blum. A PAC-style model for learning from labeled and unlabeled data. In *18th Annual Conference on Learning Theory*, 2005.

[3] M. Belkin and P. Niyogi. Semi-supervised learning on Riemannian manifolds. *Machine Learning*, 56, Invited, Special Issue on Clustering:209–239, 2004.

[4] A. Blum and T. Mitchell. Combining labeled and unlabeled data with co-training. In *11th Annual Conference on Learning Theory*, 1998.

[5] V. Castelli and T. M. Cover. The relative values of labeled and unlabeld samples in pattern recognition with an unknown mixing parameters. *IEEE Trans. Information Theory*, 42((6):2102–2117, 1996.

[6] O. Chapelle, J. Weston, and B. Scholkopf. Cluster kernels for semi-supervised learning. *NIPS*, 15, 2002.

[7] O. Chapelle and A. Zien. Semi-supervised classification by low density separation. In *10th International Workshop on Artificial Intelligence and Statistics*, 2005.

[8] S. Dasgupta, M. L. Littman, and D. McAllester. PAC generalization bounds for co-training. *NIPS*, 14, 2001.

[9] L. Devroye, L. Gyorfi, and G. Lugosi. *A probabilistic theory of pattern recognition*. Springer, New York, Berlin, Heidelberg, 1996.

[10] J. Ratsaby. The complexity of learning from a mixture of labeled and unlabeled examples. In *Phd Thesis*, 1994.

[11] J. Ratsaby and S. S. Venkatesh. Learning from a mixture of labeled and unlabeled examples with parametric side information. In *8th Annual Conference on Learning Theory*, 1995.

[12] A. B. Tsybakov. Optimal aggregation of classifiers in statistical learning. *Ann. Statist.*, 32(1):135–166, 1996.

[13] V. N. Vapnik. *Statistical Learning Theory*. Wiley, New York, 1998.

[14] Y. Yang. Minimax nonparametric classification- part I: Rates of convergence, part II: Model selection for adaptation. *IEEE Trans. Inf. Theory*, 45:2271–2292, 1999.

[15] X. Zhu. Semi-supervised literature survey. Technical Report 1530, Department of Computer Science, University of Wisconsin Madison, December 2006.

